# $(\mathbf{RF})^2$ — Random Forest Random Field

**Nadia Payet and Sinisa Todorovic**
School of Electrical Engineering and Computer Science
Oregon State University
payetn@onid.orst.edu, sinisa@eecs.oregonstate.edu

## Abstract

We combine random forest (RF) and conditional random field (CRF) into a new computational framework, called random forest random field $(\mathrm{RF})^2$. Inference of $(\mathrm{RF})^2$ uses the Swendsen-Wang cut algorithm, characterized by Metropolis-Hastings jumps. A jump from one state to another depends on the ratio of the proposal distributions, and on the ratio of the posterior distributions of the two states. Prior work typically resorts to a parametric estimation of these four distributions, and then computes their ratio. Our key idea is to instead directly estimate these ratios using RF. RF collects in leaf nodes of each decision tree the class histograms of training examples. We use these class histograms for a non-parametric estimation of the distribution ratios. We derive the theoretical error bounds of a two-class $(\mathrm{RF})^2$. $(\mathrm{RF})^2$ is applied to a challenging task of multiclass object recognition and segmentation over a random field of input image regions. In our empirical evaluation, we use only the visual information provided by image regions (e.g., color, texture, spatial layout), whereas the competing methods additionally use higher-level cues about the horizon location and 3D layout of surfaces in the scene. Nevertheless, $(\mathrm{RF})^2$ outperforms the state of the art on benchmark datasets, in terms of accuracy and computation time.

## 1   Introduction

This paper presents a new computational framework, called random forest random field $(\mathrm{RF})^2$, which provides a principled way to jointly reason about multiple, statistically dependent random variables and their attributes. We derive theoretical performance bounds of $(\mathrm{RF})^2$, and demonstrate its utility on a challenging task of conjoint object recognition and segmentation.

Identifying subimage ownership among occurrences of distinct object classes in an image is a fundamental, and one of the most actively pursued problem in computer vision, machine learning, and artificial intelligence [1–11]. The goal is to assign the label of one of multiple semantic classes to each image pixel. Our approach builds on the following common recognition strategies: (i) Labels of neighboring image parts are likely to be correlated – one of the main principles of perceptual organization; and (ii) Recognized objects dictate which other objects to expect in the scene, and their scale and spatial configuration – one of the main principles of context-driven recognition that "binds" all object detections in a coherent scene interpretation. We formalize perceptual grouping and context by a graphical model aimed at capturing statistical dependencies among random variables (i.e., labels or attributes) associated with different pixel neighborhoods. Thus, we derive a unified framework for combined object recognition and segmentation, as a graph-structured prediction of all random variables in a single, consistent model of the scene.

The graphical model we use is Conditional Random Field (CRF) [12]—one of the most popular models for structured inference over pixels [2, 3], patches [4, 5], or image regions [6–8], for object recognition and segmentation. CRF defines a posterior distribution of hidden random variables $\boldsymbol{Y}$ (labels), given observed image features $\boldsymbol{X}$, in a factored form: $p(\boldsymbol{Y}|\boldsymbol{X};\boldsymbol{\theta}) = \frac{1}{Z(\boldsymbol{\theta})}\prod_c \psi_c(\boldsymbol{Y}_c, \boldsymbol{X}; \boldsymbol{\theta})$.

Each potential $\psi_c$ is a function over a subset $\boldsymbol{Y}_c \subseteq \boldsymbol{Y}$, conditioned on $\boldsymbol{X}$, and parameterized by $\boldsymbol{\theta}$. The potentials are often defined as linear functions of parameters, $\psi_c(\boldsymbol{Y}_c, \boldsymbol{X}; \boldsymbol{\theta}) = \boldsymbol{\theta}^{\mathrm{T}} \boldsymbol{\Psi}_c$, where $\boldsymbol{\Psi}_c$ is the output of some detectors over observables $\boldsymbol{X}$ [2–4]. This means that $p(\boldsymbol{Y}|\boldsymbol{X}; \boldsymbol{\theta})$ is modeled as a log-linear function, which is not adequate when the detector outputs do not provide a linear separability of the classes. Learning $\boldsymbol{\theta}$ is hard, because computation of the partition function $Z(\boldsymbol{\theta})$ is intractable for most graphs (except for chains and trees). Inference is typically posed as the joint MAP assignment that minimizes the energy $\sum_c \psi_c(\boldsymbol{Y}_c, \boldsymbol{X}; \boldsymbol{\theta})$, which is also intractable for general graphs. The intractability of CRF learning and inference often motivates prior work to resort to *approximate* algorithms, e.g., graph-cuts, and loopy belief propagation (LBP). The effect of these approximations on the original semantics of CRF is poorly understood. For example, an approximate inference stuck in a local maximum may not represent the intended consistent scene interpretation.

**Motivation:** Some of the aforementioned shortcomings can be addressed when CRF inference is conducted using the Metropolis-Hastings (MH) algorithm. MH draws samples $\boldsymbol{Y}^{(t)}$ from the CRF's posterior, $p(\boldsymbol{Y}|\boldsymbol{X})$, and thus generates a Markov chain in which state $\boldsymbol{Y}^{(t+1)}$ depends only on the previous state $\boldsymbol{Y}^{(t)}$. The jumps between the states are reversible, and governed by a proposal density $q(\boldsymbol{Y}^{(t)} \to \boldsymbol{Y}^{(t+1)})$. The proposal is accepted if the acceptance rate, $\alpha$, drawn from $U(0,1)$, satisfies $\alpha < \min\{1, \frac{q(\boldsymbol{Y}^{(t+1)} \to \boldsymbol{Y}^{(t)})}{q(\boldsymbol{Y}^{(t)} \to \boldsymbol{Y}^{(t+1)})} \frac{p(\boldsymbol{Y}^{(t+1)}|\boldsymbol{X})}{p(\boldsymbol{Y}^{(t)}|\boldsymbol{X})}\}$. MH provides strong theoretical guarantees of convergence to the globally optimal state. As can be seen, the entire inference process is regulated by *ratios* of the proposal and posterior distributions. Consequently, the bottleneck of every CRF learning and inference — namely, computing the partition function $Z$ — is eliminated in MH.

Our key idea is to directly estimate the ratios of the proposal and posterior distributions, instead of computing each individual distribution for conducting MH jumps. Previous work on MH for CRFs usually commits to linear forms of the potential functions, and spends computational resources on estimating the four distributions: $q(\boldsymbol{Y}^{(t+1)} \to \boldsymbol{Y}^{(t)})$, $q(\boldsymbol{Y}^{(t)} \to \boldsymbol{Y}^{(t+1)})$, $p(\boldsymbol{Y}^{(t+1)}|\boldsymbol{X})$ and $p(\boldsymbol{Y}^{(t)}|\boldsymbol{X})$. In contrast, our goal is to directly estimate the two ratios, $\frac{q(\boldsymbol{Y}^{(t+1)} \to \boldsymbol{Y}^{(t)})}{q(\boldsymbol{Y}^{(t)} \to \boldsymbol{Y}^{(t+1)})}$ and $\frac{p(\boldsymbol{Y}^{(t+1)}|\boldsymbol{X})}{p(\boldsymbol{Y}^{(t)}|\boldsymbol{X})}$, in a non-parametric manner, since the acceptance rate of MH jumps depends only on these ratios. To this end, we use the random forests (RF) [13]. Given a training set of labeled examples, RF grows many decision trees. We view the trees as a way of discriminatively structuring evidence about the class distributions in the training set. In particular, each leaf of each tree in RF stores a histogram of the number of training examples from each class that reached that leaf. When a new example is encountered, it is "dropped" down each of the trees in the forest, until it reaches a leaf in every tree. The class histograms stored in all these leaves can then be used as a robust estimate of the ratio of that example's posterior distributions. This is related to recent work on Hough forests for object detection and localization [14], where leaves collect information on locations and sizes of bounding boxes of objects in training images. However, they use this evidence to predict a spatial distribution of bounding boxes in a test image, whereas we use the evidence stored in tree leaves to predict the distribution *ratios*. Evidence trees are also used in [15], but only as a first stage of a stacked-classifier architecture which replaces the standard majority voting of RF.

RF is difficult to analyze [13, 16]. Regarding consistency of RF, it is known that their rate of convergence to the optimal Bayes' rule depends only on the number of informative variables. It is also shown that RF that cuts down to pure leaves uses a weighted, layered, nearest neighbor rule [16]. We are not aware of any theoretical analysis of RF as an estimator of ratios of posterior distributions.

**Contributions:** We combine RF and CRF into a new, principled and elegant computational framework $(\mathrm{RF})^2$. Learning is efficiently conducted by RF which collects the class histograms of training examples in leaf nodes of each decision tree. This evidence is then used for the non-parametric estimation of the ratios of the proposal and posterior distributions, required by MH-based inference of $(\mathrm{RF})^2$. We derive the theoretical error bounds of estimating distribution ratios by a two-class RF, which is then used to derive the theoretical performance bounds of a two-class $(\mathrm{RF})^2$.

**Paper Organization:** Sections 2–4 specify the CRF model, its MH-based inference, and RF-based learning. Sections 5–6 present our experimental evaluation, and theoretical analysis of $(\mathrm{RF})^2$.

## 2 CRF Model

We formulate multiclass object recognition and segmentation as the MAP inference of a CRF, defined over a set of multiscale image regions. Regions are used as image features, because they are dimensionally matched with 2D object occurrences in the image, and thus facilitate modeling of various perceptual-organization and contextual cues (e.g., continuation, smoothness, containment, and adjacency) that are often used in recognition [6–11]. Access to regions is provided by the state-of-the-art, multiscale segmentation algorithm of [17], which detects and closes object (and object-part) boundaries using the domain knowledge. Since the right scale at which objects occur is unknown, we use all regions from all scales.

The extracted regions are organized in a graph, $G = (V, E)$, with $V$ and $E$ are sets of nodes and edges. The nodes $i=1,\ldots,N$ correspond to multiscale segments, and edges $(i,j) \in E$ capture their spatial relations. Each node $i$ is characterized by a descriptor vector, $x_i$, whose elements describe photometric and geometric properties of the corresponding region (e.g., color, shape, filter responses). A pair of regions can have one of the following relationships: (1) ascendent/descendent, (2) touching, and (3) far. Since the segmentation algorithm of [17] is strictly hierarchical, region $i$ is descendent of region $j$, if $i$ is fully embedded as subregion within ancestor $j$. Two regions $i$ and $j$ touch if they share a boundary part. Finally, if $i$ and $j$ are not in the hierarchical and touch relationships then they are declared as far. Edges connect all node pairs $E = V \times V$, $|E| = N^2$. Each edge $(i,j)$ is associated with a tag, $e_{ij}$, indicating the relationship type between $i$ and $j$.

CRF is defined as the graphical model over $G$. Let $\boldsymbol{Y} = \{y_i\}$ denote all random variables associated with the nodes, indicating the class label of the corresponding region, $y_i \in \{0, 1, \ldots, K\}$, where $K$ denotes the total number of object classes, and label 0 is reserved for the background class. Let $p_i = p(y_i|x_i)$ and $p_{ij} = p(y_i, y_j|x_i, x_j, e_{ij})$ denote the posterior distributions over nodes and pairs of nodes. Then, we define CRF as

$$p(\boldsymbol{Y}|G) = \prod_{i \in V} p(y_i|x_i) \prod_{(i,j) \in E} p(y_i, y_j|x_i, x_j, e_{ij}) = \prod_{i \in V} p_i \prod_{(i,j) \in E} p_{ij} \,. \qquad (1)$$

Multi-coloring of CRF is defined as the joint MAP assignment $\boldsymbol{Y}^* = \arg\max_{\boldsymbol{Y}} p(\boldsymbol{Y}|G)$. In the following section, we explain how to conduct this inference.

## 3 CRF Inference

For CRF inference, we use the Swendsen-Wang cut algorithm (SW-cut), presented in [18]. SW-cut iterates the Metropolis-Hastings (MH) reversible jumps through the following two steps. (1) Graph clustering: SW-cut probabilistically samples connected components, $CC$'s, where each $CC$ represents a subset of nodes with the same color. This is done by probabilistically cutting edges between all graph nodes that have the same color based on their posterior distributions $p_{ij} = p(y_i, y_j|x_i, x_j, e_{ij})$. (2) Graph relabeling: SW-cut randomly selects one of the $CC$'s obtained in step (1), and randomly flips the color of all nodes in that $CC$, and cuts their edges with the rest of the graph nodes having that same color. In each iteration, SW-cut probabilistically decides whether to accept the new coloring of the selected $CC$, or to keep the previous state. Unlike other MCMC methods that consider one node at a time (e.g., Gibbs sampler), SW-cut operates on a number of nodes at once. Consequently, SW-cut converges faster and enables inference on relatively large graphs. Below, we review steps (1) and (2) of SW-cut, for completeness.

In step (1), edges of $G$ are probabilistically sampled. This re-connects all nodes into new connected components $CC$. If two nodes $i$ and $j$ have different labels, they cannot be in the same $CC$, so their edge remains intact. If $i$ and $j$ have the same label, their edge is probabilistically sampled according to posterior distribution $p_{ij}$. If in the latter case edge $(i,j)$ is not sampled, we say that it has been probabilistically "cut". Step (1) results in a state $A$. In step (2), we choose at random a connected component $CC$ from step (1), and randomly reassign a new color to all nodes in that $CC$. To separate the re-colored $CC$ from the rest of the graph, we cut existing edges that connect $CC$ to the rest of the graph nodes with that same color. Step (2) results in a new state $B$. SW-cut accepts state $B$ if the acceptance rate is sufficiently large via a random thresholding. Let $q(A \rightarrow B)$ be the proposal probability for moving from state $A$ to $B$, and let $q(B \rightarrow A)$ denote the converse. The acceptance rate, $\alpha(A \rightarrow B)$, of the move from $A$ to $B$ is defined as

$$\alpha(A \rightarrow B) = \min\left(1, \frac{q(B \rightarrow A)p(\boldsymbol{Y} = B|G)}{q(A \rightarrow B)p(\boldsymbol{Y} = A|G)}\right). \qquad (2)$$

The computation complexity of each move is relatively low. The ratio $\frac{q(B \to A)}{q(A \to B)}$ in (2) involves only those edges that are "cut" around $CC$ in states $A$ and $B$ – not all edges. Also, the ratio $\frac{p(\boldsymbol{Y} = B|G)}{p(\boldsymbol{Y} = A|G)}$ accounts only for the recolored nodes in $CC$ – not the entire graph $G$, since all other probabilities have not changed from state $A$ to state $B$. Thus, from Eq. (1), the ratios of the proposal and posterior distributions characterizing states $A$ and $B$ can be specified as

$$\frac{q(B \to A)}{q(A \to B)} = \frac{\prod_{(i,j) \in \text{Cut}_B}(1 - p_{ij}^B)}{\prod_{(i,j) \in \text{Cut}_A}(1 - p_{ij}^A)}, \quad \text{and} \quad \frac{p(\boldsymbol{Y} = B|G)}{p(\boldsymbol{Y} = A|G)} = \prod_{i \in CC} \frac{p_i^B}{p_i^A} \cdot \prod_{j \in \mathcal{N}(i)} \frac{p_{ij}^B}{p_{ij}^A}. \tag{3}$$

where $\text{Cut}_A$ and $\text{Cut}_B$ denote the sets of "cut" edges in states $A$ and $B$, and $\mathcal{N}(i)$ is the set of neighbors of node $i$, $\mathcal{N}(i) = \{j : j \in V, (i,j) \in E\}$.

As shown in [18], SW-cut is relatively insensitive to different initializations. In our experiments, we initialize all nodes in the CRF with label 0. Next, we show how to compute the ratios in Eq. (3).

## 4   Learning

RF can be used for estimating the ratios of the proposal and posterior distributions, given by Eq. (3), since RF provides near Bayesian optimal decisions, as theoretically shown by Breiman [13]. In the following, we describe how to build RF, and use it for computing the ratios in Eq. (3).

Our training data represent a set of $M$ labeled regions. If region $i$ falls within the bounding box of an object in class $y \in \{1, 2, \ldots, K\}$, it receives label $y$. If $i$ covers a number of bounding boxes of different classes then $i$ is added to the training set multiple times to account for all distinct class labels it covers. Each region $i$ is characterized by a d-dimensional descriptor vector, $x_i \in \mathbb{R}^d$, which encodes the photometric and geometric properties of $i$. The training dataset $\{(x_i, y_i) : i = 1, \ldots, M\}$ is used to learn an ensemble of $T$ decision trees representing RF.

In particular, each training sample is passed through every decision tree from the ensemble until it reaches a leaf node. Each leaf $l$ records a class histogram, $\boldsymbol{\Phi}_l = \{\phi_l(y) : y = 1, \ldots, K\}$, where $\phi_l(y)$ counts the number of training examples belonging to class $y$ that reached $l$. The total number of training examples in $l$ is then $\|\boldsymbol{\Phi}_l\|$. Also, for each pair of leaves $(l, l')$, we record a two-class histogram, $\boldsymbol{\Psi}_{ll'} = \{\psi_{ll'}(y, y', e) : y, y' = 1, \ldots, K; e = 1, 2, 3\}$, where $\psi_{ll'}(y, y', e)$ counts the number of pairs of training examples belonging to classes $y$ and $y'$ that reached leaves $l$ and $l'$, and also have the relationship type $e$ – namely, ascendent/descendent, touching, or far relationship.

Given $\boldsymbol{\Phi}_l$ and $\boldsymbol{\Psi}_{ll'}$, we in a position to estimate the ratios of the proposal and posterior distributions, defined in (3), which control the Metropolis-Hastings jumps in the SW-cut. Suppose two regions, represented by their descriptors $x_i$ and $x_j$, are labeled as $y_i^A$ and $y_j^A$ in state A, and $y_i^B$ and $y_j^B$ in state $B$ of one iteration of the SW-cut. Also, after passing $x_i$ and $x_j$ through $T$ decision trees of the learned RF, suppose they reached leaves $l_i^t$ and $l_j^t$ in each tree $t = 1, \ldots, T$. Then, we compute

$$\frac{p_i^B}{p_i^A} = \frac{\sum_{t=1}^T \phi_{l_i^t}(y_i^B)}{\sum_{t=1}^T \phi_{l_i^t}(y_i^A)}, \qquad \frac{p_{ij}^B}{p_{ij}^A} = \frac{\sum_{t=1}^T \psi_{l_i^t l_j^t}(y_i^B, y_j^B, e_{ij})}{\sum_{t=1}^T \psi_{l_i^t l_j^t}(y_i^A, y_j^A, e_{ij})}, \quad \text{for estimating} \quad \frac{p(\boldsymbol{Y} = B|G)}{p(\boldsymbol{Y} = A|G)}. \tag{4}$$

To estimate the ratio of the proposal distributions, $\frac{q(B \to A)}{q(A \to B)}$, it is necessary to compute each individual probability $p_{ij}$, since the nominator and denominator of $\frac{q(B \to A)}{q(A \to B)}$ do not contain the same set of "cut" edges, $\text{Cut}_A \neq \text{Cut}_B$, as specified in (3). Thus, we compute

$$p_{ij} = \frac{\sum_{t=1}^T \psi_{l_i^t l_j^t}(y_i, y_j, e_{ij})}{\sum_{t=1}^T \|\boldsymbol{\Phi}_{l_i^t}\| \|\boldsymbol{\Phi}_{l_j^t}\|} \quad \text{for estimating} \quad \frac{q(B \to A)}{q(A \to B)}. \tag{5}$$

In the following, we first present our empirical evaluation of $(\text{RF})^2$, and then derive the theoretical performance bounds of a simple, two-class $(\text{RF})^2$.

## 5   Results

$(\text{RF})^2$ is evaluated on the task of object recognition and segmentation on two benchmark datasets. First, the MSRC dataset consists of 591 images showing objects from 21 categories [3]. We use the

standard split of MSRC into training and test images [3]. Second, the Street-Scene dataset consists of 3547 images of urban environments, and has manually annotated regions [6, 19]. As in [6], one fifth of the Street-Scene images are used for testing, and the rest, for training. Both datasets provide labels of bounding boxes around object occurrences as ground truth.

Images are segmented using the multiscale segmentation algorithm of [17], which uses the perceptual significance of a region boundary, $P_b \in [0, 100]$, as an input parameter. We vary $P_b = 30{:}10{:}150$, and thus obtain a hierarchy of regions for each image. A region is characterized by a descriptor vector consisting of the following properties: (i) 30-bin color histogram in the CIELAB space; (ii) 250-dimensional histogram of filter responses of the MR8 filter bank, and the Laplacian of Gaussian filters computed at each pixel, and mapped to 250 codewords whose dictionary is obtained by K-means over all training images; (iii) 128-dimensional region boundary descriptor measuring oriented contour energy along 8 orientations of each cell of a $4 \times 4$ grid overlaid over the region's bounding box; (iv) coordinates of the region's centroid normalized to the image size. Regions extracted from training images are used for learning RF. A region that falls within a bounding box is assigned the label of that box. If a region covers a number of bounding boxes of different classes, it is added to the training set multiple times to account for each distinct label. We use the standard random splits of training data to train 100 decision trees of RF, constructed in the top-down way. The growth of each tree is constrained so its depth is less than 30, and its every leaf node contains at least 20 training examples. To recognize and segment objects in a new test image, we first extract a hierarchy of regions from the image by the segmentation algorithm of [17]. Then, we build the fully connected CRF graph from the extracted regions (Sec. 2), and run the SW-cut inference (Sec. 4).

We examine the following three variants of $(\text{RF})^2$: $(\text{RF})^2$-1 — The spatial relationships of regions, $e_{ij}$, are not accounted for when computing $p_{ij}$ in Eq. (4) and Eq. (5); $(\text{RF})^2$-2 — The region relationships touching and far are considered, while the ascendent/descendent relationship is not captured; and $(\text{RF})^2$-3 — All three types of region layout and structural relationships are modeled. In this paper, we consider $(\text{RF})^2$-3 as our default variant, and explicitly state when the other two are used instead. Note that considering region layouts and structure changes only the class histograms recorded by leaf nodes of the learned decision trees, but it does not increase complexity.

For quantitative evaluation, we compute the pixel-wise classification accuracy averaged across all test images, and object classes. This metric is suitable, because it does not favor object classes that occur in images more frequently. Tab. 1 and Tab. 2 show our pixel-wise classification accuracy on MSRC and Street-Scene images. Table. 2 also compares the three variants of $(\text{RF})^2$ on MSRC and Street-Scene images. The additional consideration of the region relationships touching and far increases performance relative to that of $(\text{RF})^2$-1, as expected. Our performance is the best when all three types of region relationships are modeled. The tables also present the pixel-wise classification accuracy of the state of the art CRF models [3,6,20,21]. Note that the methods of [6,21] additionally use higher-level cues about the horizon location and 3D scene layout in their object recognition and segmentation. As can be seen, $(\text{RF})^2$ outperforms the latest CRF models on both datasets.

Our segmentation results on example MSRC and Street-Scene images are shown in Fig. 5. Labels of the finest-scale regions are depicted using distinct colors, since pixels get labels of the finest-scale regions. As can be seen, $(\text{RF})^2$ correctly identifies groups of regions that belong to the same class.

Since the depth of each decision tree in RF is less than 30, the complexity of dropping an instance through one tree is $O(1)$, and through RF with $T$ trees is $O(T)$. Our C-implementation of the RF-

| Method | Aeroplane | Bicycle | Bird | Boat | Body | Book | Building | Car | Cat | Chair | Cow | Dog | Face | Flower | Grass | Road | Sheep | Sign | Sky | Tree | Water |
|---|---|---|---|---|---|---|---|---|---|---|---|---|---|---|---|---|---|---|---|---|---|
| [10] | 88 | 91 | 34 | 49 | 54 | 93 | 30 | 82 | 56 | 74 | 68 | 54 | 77 | 90 | 71 | 31 | 64 | 82 | 84 | 69 | 58 |
| [22] | 82 | 72 | 24 | 18 | 66 | 93 | 49 | 74 | 75 | 51 | 97 | 35 | 87 | 74 | 88 | 78 | 97 | 36 | 78 | 79 | 54 |
| [23] | 83 | 79 | 30 | 27 | 67 | 80 | 69 | 70 | 68 | 45 | 78 | 52 | 84 | 47 | 96 | 78 | 80 | 61 | 95 | 87 | 67 |
| [20] | **100** | 98 | 11 | 63 | 55 | 78 | 73 | **88** | 11 | **80** | 74 | 43 | 72 | 72 | 96 | 76 | 90 | 92 | 50 | 76 | 61 |
| [3] | 60 | 75 | 19 | 7 | 62 | 92 | 62 | 63 | 54 | 15 | 58 | 19 | 74 | 63 | 97 | **86** | 50 | 35 | 83 | 86 | 53 |
| **Ours** | **100** | **99** | **42** | **69** | **68** | **95** | **74** | **88** | **77** | **80** | **99** | **61** | **91** | **93** | **99** | 78 | **99** | **93** | **96** | **90** | **68** |

Table 1: The average pixel-wise classification accuracy on the MSRC dataset. $(\text{RF})^2$ yields the best performance for all object classes except one.

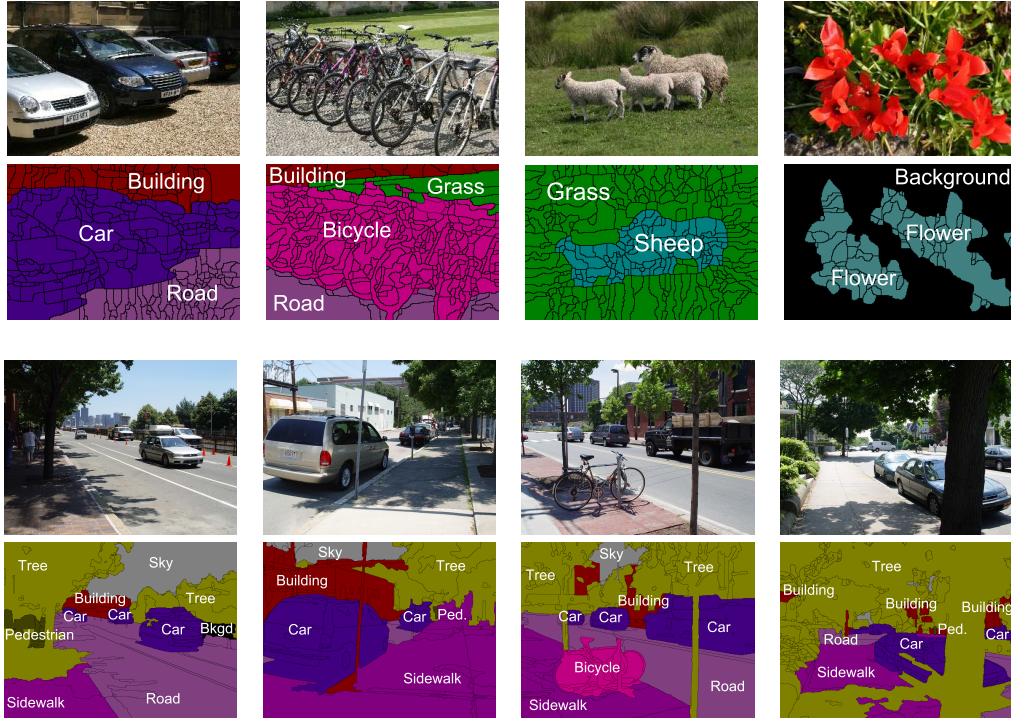

Figure 1: Our object recognition and segmentation results on example images from the MSRC dataset (top two rows), and the Street-Scene dataset (bottom two rows). The figure depicts boundaries of the finest-scale regions found by the multiscale algorithm of [17], and the color-coded labels of these regions inferred by $(\text{RF})^2$. The results are good despite the presence of partial occlusion, and changes in illumination and scale. (best viewed in color)

| Method | MSRC | StreetScene | Test time |
|---|---|---|---|
| $(\text{RF})^2$-1 | 69.5%±13.7% | 78.2%±0.5% | 45s |
| $(\text{RF})^2$-2 | 80.2%±14.4% | 86.7%±0.5% | 31s |
| $(\text{RF})^2$-3 | 82.9%±15.8% | 89.8%±0.6% | 31s |
| [20] | 70.0% | N/A | N/A |
| [21] | 76.4% | 83.0% | N/A |
| [6] | N/A | 84.2% | N/A |
| [3] | 70.0% | N/A | 10-30s |

Table 2: The average pixel-wise classification accuracy and average computation times on the MSRC and Street-Scene datasets of the three variants of our approach with those of the state-of-the-art CRF-based methods.

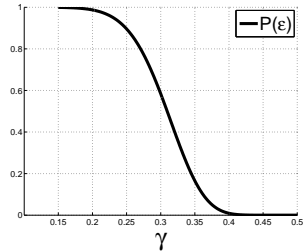

Figure 2: The probability of classification error of $(\text{RF})^2$, $P(\epsilon)$, given by Eq. (6) and Theorem 1 as a function of the margin, $\gamma$, of RF.

guided SW-cut inference of CRF takes 10s–30s on a 2.40GHz PC with 3.48GB RAM for MSRC and Street-Scene images. Table 2 shows that our average running times are comparable to those of the other CRF methods that use approximate inference [3, 6, 20, 21].

## 6   Theoretical Analysis

We are interested in a theoretical explanation of the good performance of $(\text{RF})^2$ presented in the previous section. In particular, we derive the theoretical performance bounds of a two-class $(\text{RF})^2$, for simplicity. As explained in Sec. 3, we use the SW-cut for $(\text{RF})^2$ inference. The SW-cut iterates the Metropolis-Hastings (MH) reversible jumps, and thus explores the state-space of solutions. An MH jump between states $A$ and $B$ is controlled by the acceptance rate $\alpha(A{\rightarrow}B)$ which depends on

the ratios of the proposal and posterior distributions, $\frac{q(B \to A)p(\boldsymbol{Y}=B|G)}{q(A \to B)p(\boldsymbol{Y}=A|G)}$. Below, we show that the error made by the two-class RF in estimating these ratios is bounded. Our derivation of the error bounds of RF is based on the theoretical analysis of evidence trees, presented in [15].

## 6.1 An Upper Error Bound of $(\mathbf{RF})^2$

An error occurs along MH jumps when a *balanced* reversible jump is encountered, i.e., when there is no preference between jumping from state $A$ to state $B$ and reverse, $\frac{q(B \to A)}{q(A \to B)}{=}1$, and RF *wrongly* predicts that the posterior distribution of state $B$ is larger than that of $A$, $\frac{p(\boldsymbol{Y}=B|G)}{p(\boldsymbol{Y}=A|G)}{\geq}1$. In this case, $\alpha(A{\to}B){=}1$, and the SW-cut will erroneously visit state $B$. We are interested in finding the probability of this error, specified as

$$P(\epsilon) = P\left(\frac{p(\boldsymbol{Y} = B|G)}{p(\boldsymbol{Y} = A|G)} \geq 1\right) = P\left(\prod_{i \in CC} \frac{p_i^B}{p_i^A} \cdot \prod_{j \in \mathcal{N}(i)} \frac{p_{ij}^B}{p_{ij}^A} \geq 1\right). \tag{6}$$

From Eq. (6), $P(\epsilon)$ can be computed using the probability density function of a product of random variables $Z_i = p_i^B/p_i^A \in [0,\infty)$, and $W_{ij} = p_{ij}^B/p_{ij}^A \in [0,\infty)$, within a specific connected component $CC$, where $|CC|{=}n$, $i = 1,\ldots,n$, and $j \in \mathcal{N}(i)$. As we will prove in the sequel, all random variables $Z_i$ have the same exponential distribution $f_{Z_i}(z){=}\lambda_1 \exp(-\lambda_1 z)$. Also, we will prove that all random variables $W_{ij}$ have the same exponential distribution $f_{W_{ij}}(w){=}\lambda_2 \exp(-\lambda_2 w)$. Then, it follows that the product $Z{=}\prod_{i=1}^n Z_i{=}(Z_i)^n$ has the distribution $f_Z(z){=}\frac{\lambda_1}{n} z^{\frac{1-n}{n}} \exp(-\lambda_1 z^{\frac{1}{n}})$. Also, the product $W{=}\prod_{i=1}^n \prod_{j \in \mathcal{N}(i)} W_{ij}{=}(W_{ij})^{nk}{\approx}(W_{ij})^n$ has the distribution $f_W(w){=}\frac{\lambda_2}{n} w^{\frac{1-n}{n}} \exp(-\lambda_2 w^{\frac{1}{n}})$, where we approximate that the number of edges within $CC$ is the same as the number of nodes in $CC$, as a result of the probabilistic "cutting" of graph edges by the SW-cut algorithm. Given $f_Z(z)$ and $f_W(w)$, from Eq. (6), we analytically derive the probability that $(\text{RF})^2$ makes a wrong prediction, $P(\epsilon) = P(Z \cdot W \geq 1)$, as stated in the following theorem.

**Theorem 1.** *The probability that $(RF)^2$ makes a wrong prediction is $P(\epsilon){=}P(Z{\cdot}W \geq 1){=}\lambda K_1(\lambda)$, where $Z{\in}[0,\infty)$ and $W{\in}[0,\infty)$ are random variables characterized by the probability density functions $f_Z(z){=}\frac{\lambda_1}{n} z^{\frac{1-n}{n}} \exp(-\lambda_1 z^{\frac{1}{n}})$ and $f_W(w){=}\frac{\lambda_2}{n} w^{\frac{1-n}{n}} \exp(-\lambda_2 w^{\frac{1}{n}})$, with parameters $\lambda_1$ and $\lambda_2$, and where $K_1$ is the modified Bessel function of the second kind, and $\lambda = 2\sqrt{\lambda_1 \lambda_2}$.*

**Proof.** Define $H = Z \cdot W$. Then, $f_H(h){=}\int_0^\infty \frac{1}{z} f_Z(z) f_W(\frac{h}{z}) dz = \frac{\lambda^2}{2n} h^{\frac{1-n}{n}} K_0(\lambda h^{\frac{1}{2n}})$, where $K_0$ is the modified Bessel function of the second kind. It follows that $P(\epsilon) = P(H{\geq}1) = 1{-}\int_0^1 f_H(h) dh = \lambda K_1(\lambda)$.□

As we will show in the following section, the parameter $\lambda$ is directly proportional to a measure of accuracy of RF predictions, referred to as probabilistic margin. Since $K_1(\lambda)$ is a decreasing function, it follows that the probability that $(\text{RF})^2$ makes a wrong prediction is upper bounded, and decreases as the probabilistic margin of RF increases.

## 6.2 A Mathematical Model of RF Performance

In this section, we derive that the RF estimates of the ratios of posteriors $Z_i$ and $W_{ij}$ have the exponential distribution. We consider a binary classification problem, for simplicity, where training and test instances may have positive and negative labels. We assume that the two classes are balanced $P(y{=}+1) = P(y{=}-1) = 1/2$. We define $\pi$ to be a fraction of pairs of instances that have certain relationship, corresponding to a particular spatial or structural relationship between pairs of regions, defined in Sec. 2. The learning algorithm that creates RF is not modeled. Instead, we assume that the learned decision trees have the following properties. Each leaf node of a decision tree: (i) stores a total of $C$ training instances that reach the leaf; and (ii) has a probabilistic margin $\gamma \in [0, 1/2)$. By margin, we mean that in every leaf reached by $C$ training instances a fraction of $1/2 + \gamma$ of the training instances will belong to one class (e.g., positive), and fraction $1/2 - \gamma$ of them will belong to the other class (e.g., negative). We say that a leaf is positive if a majority of the training instances collected by the leaf is positive, or otherwise, we say that the leaf is negative. It is straightforward to show that when a positive instance is dropped through one of the decision trees in RF, it will

reach a positive leaf with probability $1/2 + \gamma$, and a negative leaf with probability $1/2 - \gamma$ [15]. Similarly holds for negative instances. A new test instance is classified by dropping it through $T$ decision trees, and taking a majority vote of the labels of all $C \cdot T$ training instances stored in the leaves reached by the test instance. We refer to this classification procedure as evidence voting [15], as opposed to decision voting over the leaf labels in the standard RF [13]. The following proposition states that the probability that evidence voting misclassifies an instance, $P(\epsilon_1)$, is upper bounded.

**Proposition 1.** *The probability that RF with $T$ trees, where every leaf stores $C$ training instances, incorrectly classifies an instance is upper bounded, $P(\epsilon_1) \leq \exp(-8CT\gamma^4)$.*

**Proof.** Evidence voting for labeling an instance can be formalized as drawing a total of $C \cdot T$ independent Bernoulli random variables, with the success rate $p_1$, whose outcomes are $\{-1, +1\}$, where $+1$ is received for correct, and $-1$ for incorrect labeling of the instance. Let $S_1$ denote a sum of these Bernoulli random variables. Thus, a positive instance is incorrectly labeled if $S_1 \leq 0$, and a negative instance is misclassified if $S_1 > 0$. Since the two classes are balanced, by applying the standard Chernoff bound, we obtain $P(\epsilon_1) = P(S_1 \leq 0) \leq \exp\left[-2CT(p_1 - 1/2)^2\right]$. The success rate $p_1$ can be derived as follows. When a positive (negative) instance is dropped through a decision tree, it will fall in a positive (negative) leaf with probability $1/2 + \gamma$, where it will be labeled as positive (negative) with probability $1/2 + \gamma$; else, the positive (negative) instance will be routed to a negative (positive) leaf with probability $1/2 - \gamma$, where it will be labeled as positive (negative) with probability $1/2 - \gamma$. Consequently, the probability that an instance is correctly labeled, i.e., the success rate of the associated Bernoulli random variable, is $p_1 = (1/2 + \gamma)(1/2 + \gamma) + (1/2 - \gamma)(1/2 - \gamma) = 1/2 + 2\gamma^2$.□

Evidence voting is also used for labeling pairs of instances. The probability that evidence voting misclassifies a pair of test instances, $P(\epsilon_2)$, is upper bounded, as stated in Proposition 2.

**Proposition 2.** *Given RF as in Proposition 1, the probability that RF incorrectly labels a pair of instances having a certain relationship is upper bounded, $P(\epsilon_2) \leq \exp(-8C^2T\pi^4\gamma^8)$.*

**Proof.** Evidence voting for labeling a pair of instances can be formalized as drawing a total of $C^2T$ independent Bernoulli random variables, with success rate $p_2$, whose outcomes are $\{-1, +1\}$, where $+1$ is received for correct, and $-1$ for incorrect labeling of the instance pair. Let $S_2$ denote a sum of these Bernoulli random variables. Then, $P(\epsilon_2) = P(S_2 \leq 0) \leq \exp\left[-2C^2T(p_2 - 1/2)^2\right]$. Similar to the proof of Proposition 1, by considering three possible cases of correct labeling of a pair of instances when dropping the pair through a decision tree, the success rate $p_2$ can be derived as $p_2 = \pi(1/2 + \gamma^2)(1/2 + \pi\gamma^2) + \pi(1/2 - \gamma^2)(1/2 - \pi\gamma^2) + (1 - \pi)(1/2) = 1/2 + 2\pi^2\gamma^4$, where $\pi$ is a fraction of pairs of instances that have the same type of relationship.□

From Proposition 1, it follows that the probability that RF makes a wrong prediction about the posterior ratio of an instance is upper bounded, $P(Z_i \geq 1) = P(\epsilon_1) = \exp(-8CT\gamma^4), \forall i \in CC$. This gives the probability density function $f_{Z_i}(z) = \lambda_1 \exp(-\lambda_1 z)$, where $\lambda_1 = 8CT\gamma^4$. In addition, From Proposition 2, it follows that the probability that RF makes a wrong prediction about the posterior ratio of a pair of instances is upper bounded, $P(W_{ij} \geq 1) = P(\epsilon_2) = \exp(-8C^2T\pi^4\gamma^8)$, $\forall i \in CC$ and $j \in \mathcal{N}(i)$. This gives the probability density function $f_{W_{ij}}(w) = \lambda_2 \exp(-\lambda_2 w)$, where $\lambda_2 = 8C^2T\pi^4\gamma^8$. By plugging these results in Theorem 1, we complete the derivation of the upper error bound of $(\text{RF})^2$. From Theorem 1, $P(\epsilon)$ decreases when any of the following parameters increases: $C$, $T$, $\gamma$, and $\pi$. Fig. 2 shows the influence of $\gamma$ on $P(\epsilon)$, when the other parameters are fixed to their typical values: $C = 20$, $T = 100$, and $\pi = 0.1$.

## 7 Conclusion

We have presented $(\text{RF})^2$ – a framework that uses the random forest (RF) for the MCMC-based inference of a conditional random field (CRF). Our key idea is to employ RF to directly compute the ratios of the proposal and posterior distributions of states visited along the Metropolis-Hastings reversible jumps, instead of estimating each individual distribution, and thus improve the convergence rate and accuracy of the CRF inference. Such a non-parametric formulation of CRF and its inference has been demonstrated to outperform, in terms of computation time and accuracy, existing parametric CRF models on the task of multiclass object recognition and segmentation. We have also derived the upper error bounds of the two-class RF and $(\text{RF})^2$, and showed that the classification error of $(\text{RF})^2$ decreases as any of the following RF parameters increases: the number of decision trees, the number of training examples stored in every leaf node, and the probabilistic margin.

# References

[1] L.-J. Li, R. Socher, and L. Fei-Fei, "Towards total scene understanding: Classification, annotation and segmentation in an automatic framework," in *CVPR*, 2009.

[2] X. He, R. S. Zemel, and M. A. Carreira-Perpinan, "Multiscale Conditional Random Fields for image labeling," in *CVPR*, 2004, pp. 695–702.

[3] J. Shotton, J. Winn, C. Rother, and A. Criminisi, "Textonboost: Joint appearance, shape and context modeling for multi-class object recognition and segmentation," in *ECCV*, 2006, pp. 1–15.

[4] J. Verbeek and B. Triggs, "Scene segmentation with CRFs learned from partially labeled images," in *NIPS*, 2007.

[5] A. Torralba, K. P. Murphy, and W. T. Freeman, "Contextual models for object detection using boosted random fields," in *NIPS*, 2004.

[6] S. Gould, T. Gao, and D. Koller, "Region-based segmentation and object detection," in *NIPS*, 2009.

[7] A. Rabinovich, A. Vedaldi, C. Galleguillos, E. Wiewiora, and S. Belongie, "Objects in context," in *ICCV*, 2007.

[8] N. Payet and S. Todorovic, "From a set of shapes to object discovery," in *ECCV*, 2010.

[9] S. Todorovic and N. Ahuja, "Unsupervised category modeling, recognition, and segmentation in images," *IEEE TPAMI*, vol. 30, no. 12, pp. 1–17, 2008.

[10] J. J. Lim, P. Arbelaez, C. Gu, and J. Malik, "Context by region ancestry," in *ICCV*, 2009.

[11] J. Sivic, B. C. Russell, A. Zisserman, W. T. Freeman, and A. A. Efros, "Unsupervised discovery of visual object class hierarchies," in *CVPR*, 2008.

[12] J. Lafferty, A. McCallum, and F. Pereira, "Conditional random fields: Probabilistic models for segmenting and labeling sequence data," in *ICML*, 2001, pp. 282–289.

[13] L. Breiman, "Random forests," *Mach. Learn.*, vol. 45, no. 1, pp. 5–32, 2001.

[14] J. Gall and V. Lempitsky, "Class-specific hough forests for object detection," in *CVPR*, 2009.

[15] G. Martinez-Munoz, N. Larios, E. Mortensen, W. Zhang, A. Yamamuro, R. Paasch, N. Payet, D. Lytle, L. Shapiro, S. Todorovic, A. Moldenke, and T. Dietterich, "Dictionary-free categorization of very similar objects via stacked evidence trees," in *CVPR*, 2009.

[16] Y. Lin and Y. Jeon, "Random forests and adaptive nearest neighbors," *Journal of the American Statistical Association*, pp. 101–474, 2006.

[17] C. F. P. Arbelaez, M. Maire and J. Malik, "From contours to regions: An empirical evaluation," in *CVPR*, 2009.

[18] A. Barbu and S.-C. Zhu, "Graph partition by Swendsen-Wang cuts," in *ICCV*, 2003, p. 320.

[19] S. Bileschi and L. Wolf, "A unified system for object detection, texture recognition, and context analysis based on the standard model feature set," in *BMVC*, 2005.

[20] C. Galleguillos, B. McFee, S. Belongie, and G. R. G. Lanckriet, "Multi-class object localization by combining local contextual interactions," in *CVPR*, 2010.

[21] S. Gould, R. Fulton, and D. Koller, "Decomposing a scene into geometric and semantically consistent regions," in *ICCV*, 2009.

[22] J. Shotton, M. Johnson, and R. Cipolla, "Semantic texton forests for image categorization and segmentation," in *CVPR*, 2008.

[23] Z. Tu and X. Bai, "Auto-context and its application to high-level vision tasks and 3D brain image segmentation," *IEEE TPAMI*, vol. 99, 2009.

